# Modeling High-Dimensional Discrete Data with Multi-Layer Neural Networks

**Yoshua Bengio**
*Dept. IRO*
*Université de Montréal*
Montreal, Qc, Canada, H3C 3J7
bengioy@iro.umontreal.ca

**Samy Bengio**[*]
IDIAP
CP 592, rue du Simplon 4,
1920 Martigny, Switzerland
bengio@idiap.ch

## Abstract

The curse of dimensionality is severe when modeling high-dimensional discrete data: the number of possible combinations of the variables explodes exponentially. In this paper we propose a new architecture for modeling high-dimensional data that requires resources (parameters and computations) that grow only at most as the square of the number of variables, using a multi-layer neural network to represent the joint distribution of the variables as the product of conditional distributions. The neural network can be interpreted as a graphical model without hidden random variables, but in which the conditional distributions are tied through the hidden units. The connectivity of the neural network can be pruned by using dependency tests between the variables. Experiments on modeling the distribution of several discrete data sets show statistically significant improvements over other methods such as naive Bayes and comparable Bayesian networks, and show that significant improvements can be obtained by pruning the network.

## 1 Introduction

The curse of dimensionality hits particularly hard on models of high-dimensional discrete data because there are many more possible combinations of the values of the variables than can possibly be observed in any data set, even the large data sets now common in data-mining applications. In this paper we are dealing in particular with multivariate discrete data, where one tries to build a model of the distribution of the data. This can be used for example to detect anomalous cases in data-mining applications, or it can be used to model the class-conditional distribution of some observed variables in order to build a classifier. A simple multinomial maximum likelihood model would give zero probability to all of the combinations not encountered in the training set, i.e., it would most likely give zero probability to most out-of-sample test cases. Smoothing the model by assigning the same non-zero probability for all the unobserved cases would not be satisfactory either because it would not provide much generalization from the training set. This could be obtained by using a multivariate multinomial model whose parameters $\theta$ are estimated by the maximum a-posteriori (MAP) principle, i.e., those that have the greatest probability, given the training data $D$, and using a diffuse prior $P(\theta)$ (e.g. Dirichlet) on the parameters.

A graphical model or Bayesian network [6, 5] represents the joint distribution of random variables $Z_1 \ldots Z_n$ with

$$P(Z_1 \ldots Z_n) = \prod_{i=1}^{n} P(Z_i | \text{Parents}_i)$$

where Parents$_i$ is the set of random variables which are called the **parents** of variable $i$ in the graphical model because they directly condition $Z_i$, and an arrow is drawn, in the graphical model, to $Z_i$, from each of its parents. A fully connected "left-to-right" graphical model is illustrated in Figure 1 (left), which corresponds to the model

$$P(Z_1 \ldots Z_n) = \prod_{i=1}^{n} P(Z_i | Z_1 \ldots Z_{i-1}). \tag{1}$$

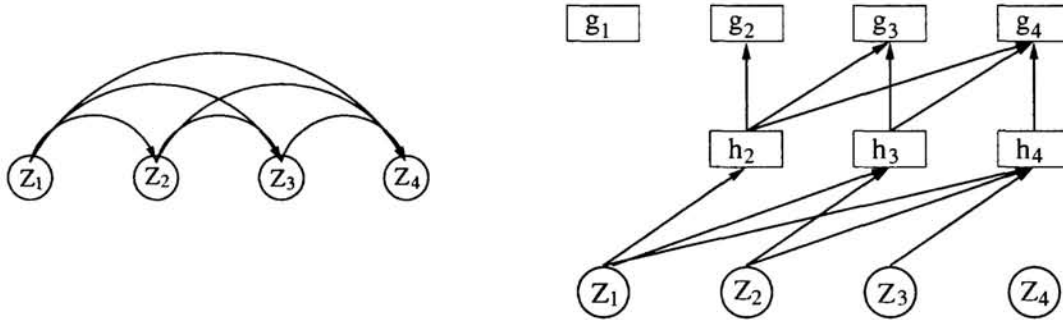

Figure 1: *Left: a fully connected "left-to-right" graphical model.*
*Right: the architecture of a neural network that simulates a fully connected "left-to-right" graphical model. The observed values $Z_i = z_i$ are encoded in the corresponding input unit group. $h_i$ is a group of hidden units. $g_i$ is a group of output units, which depend on $z_1 \ldots z_{i-1}$, representing the parameters of a distribution over $Z_i$. These conditional probabilities $P(Z_i | Z_1 \ldots Z_{i-1})$ are multiplied to obtain the joint distribution.*

Note that this representation depends on the ordering of the variables (in that all previous variables in this order are taken as parents). We call each combination of the values of Parents$_i$ a *context*. In the "exact" model (with the full table of all possible contexts) all the orders are equivalent, but if approximations are used, different predictions could be made by different models assuming different orders.

In graphical models, the curse of dimensionality shows up in the representation of conditional distributions $P(Z_i | \text{Parents}_i)$ where $Z_i$ has many parents. If $Z_j \in \text{Parents}_i$ can take $n_j$ values, there are $\prod_j n_j$ different contexts which can occur in which one would like to estimate the distribution of $Z_i$. This serious problem has been addressed in the past by two types of approaches, which are sometimes combined:

1. *Not modeling all the dependencies between all the variables*: this is the approach mainly taken with most graphical models or Bayes networks [6, 5]. The set of independencies can be assumed using a-priori or human expert knowledge or can be learned from data. See also [2] in which the set *Parents$_i$* is restricted to at most one element, which is chosen to maximize the correlation with $Z_i$.

2. *Approximating the mathematical form of the joint distribution* with a form that takes only into account dependencies of lower order, or only takes into account some of the possible dependencies, e.g., with the Rademacher-Walsh expansion or multi-binomial [1, 3], which is a low-order polynomial approximation of a full joint binomial distribution (and is used in the experiments reported in this paper).

The approach we are putting forward in this paper is mostly of the second category, although we are using simple non-parametric statistics of the dependency between pairs of variables to further reduce the number of required parameters.

In the multi-binomial model [3], the joint distribution of a set of binary variables is approximated by a polynomial. Whereas the "exact" representation of $P(Z_1 = z_1, \ldots Z_n = z_n)$ as a function of $z_1 \ldots z_n$ is a polynomial of degree $n$, it can be approximated with a lower

degree polynomial, and this approximation can be easily computed using the Rademacher-Walsh expansion [1] (or other similar expansions, such as the Bahadur-Lazarsfeld expansion [1]). Therefore, instead of having $2^n$ parameters, the approximated model for $P(Z_1, \ldots Z_n)$ only requires $O(n^k)$ parameters. Typically, order $k = 2$ is used. The model proposed here also requires $O(n^2)$ parameters, but it allows to model dependencies between tuples of variables, with more than 2 variables at a time.

In previous related work by Frey [4], a fully-connected graphical model is used (see Figure 1, left) but each of the conditional distributions is represented by a logistic, which take into account only first-order dependency between the variables:

$$P(Z_i = 1 | Z_1 \ldots Z_{i-1}) = \frac{1}{1 + \exp(-w_0 - \sum_{j<i} w_j Z_j)}.$$

In this paper, we basically extend Frey's idea to using a neural network with a hidden layer, with a particular architecture, allowing multinomial or continuous variables, and we propose to prune down the network weights. Frey has named his model a *Logistic Autoregressive Bayesian Network* or LARC. He argues that the prior variances on the logistic weights (which correspond to inverse weight decays) should be chosen inversely proportional to the number of conditioning variables (i.e. the number of inputs to the particular output neuron). The model was tested on a task of learning to classify digits from 8x8 binary pixel images. Models with different orderings of the variables were compared and did not yield significant differences in performance. When averaging the predictive probabilities from 10 different models obtained by considering 10 different random orderings, Frey obtained small improvements in likelihood but not in classification. The model performed better or equivalently to other models tested: CART, naive Bayes, K-nearest neighbors, and various Bayesian models with hidden variables (Helmholtz machines). These results are impressive, taking into account the simplicity of the LARC model.

## 2 Proposed Architecture

The proposed architecture is a "neural network" implementation of a graphical model where all the variables are observed in the training set, with the hidden units playing a significant role to share parameters across different conditional distributions. Figure 1 (right) illustrates the model in the simpler case of a fully connected (left-to-right) graphical model (Figure 1, left). The neural network represents the parametrized function

$$f_\theta(z_1, \ldots, z_n) = log(\hat{P}_\theta(Z_1 = z_1, \ldots, Z_n = z_n)) \tag{2}$$

approximating the joint distribution of the variables, with parameters $\theta$ being the weights of the neural network. The architecture has three layers, with each layer organized in **groups** associated to each of the variables. The above log-probability is computed as the sum of conditional log-probabilities

$$f_\theta(z_1, \ldots, z_n) = \sum_{i=1}^n log(P(Z_i = z_i | g_i(z_1, \ldots, z_{i-1})))$$

where $g_i(z_1, \ldots, z_{i-1})$ is the vector-valued output of the $i$-th group of output units, and it gives the value of the parameters of the distribution of $Z_i$ when $Z_1 = z_1, Z_2 = z_2, \ldots, Z_{i-1} = z_{i-1}$. For example, in the ordinary discrete case, $g_i$ may be the vector of probabilities associated with each of the possible values of the multinomial random variable $Z_i$. In this case, we have

$$P(Z_i = i'|g_i) = g_{i,i'}$$

In this example, a *softmax* output for the $i$-th group may be used to force these parameters to be positive and sum to 1, i.e.,

$$g_{i,i'} = \frac{e^{g'_{i,i'}}}{\sum_{i'} e^{g'_{i,i'}}}$$

where $g'_{i,i'}$ are linear combinations of the hidden units outputs, with $i'$ ranging over the number of elements of the parameter vector associated with the distribution of $Z_i$ (for a fixed value of $Z_1 \ldots Z_{i-1}$). To guarantee that the functions $g_i(z_1, \ldots, z_{i-1})$ only depend on $z_1 \ldots z_{i-1}$ and not on any of $z_i \ldots z_n$, the connectivity struture of the hidden units must be constrained as follows:

$$g'_{i,i'} = b_{i,i'} + \sum_{j \leq i} \sum_{j'=1}^{m_j} w_{i,i',j,j'} h_{j,j'}$$

where the $b$'s are biases and the $w$'s are weights of the output layer, and the $h_{j,j'}$ is the output of the $j'$-th unit (out of $m_j$ such units) in the $j$-th group of hidden layer nodes. It may be computed as follows:

$$h_{j,j'} = \tanh(c_{j,j'} + \sum_{k < j} \sum_{k'=1}^{n_k} v_{j,j',k,k'} z_{k,k'})$$

where the $c$'s are biases and the $v$'s are the weights of the hidden layer, and $z_{k,k'}$ is $k'$-th element of the vectorial input representation of the value $Z_k = z_k$. For example, in the binary case ($z_i = 0$ or 1) we have used only one input node, i.e.,

$$Z_i \ binomial \ \rightarrow z_{i,0} = z_i$$

and in the multinomial case we use the one-hot encoding,

$$Z_i \in \{0, 1, \ldots n_i - 1\} \ \rightarrow z_{i,i'} = \delta_{z_i,i'}$$

where $\delta_{i,i'} = 1$ if $i = i'$ and 0 otherwise. The input layer has $n - 1$ groups because the value $Z_n = z_n$ is not used as an input. The hidden layer also has $n - 1$ groups corresponding to the variables $j = 2$ to $n$ (since $P(Z_1)$ is represented unconditionally in the first output group, its corresponding group does not need any hidden units or inputs, but just has biases).

## 2.1 Discussion

The number of free parameters of the model is $O(n^2 H)$ where $H = \max_i m_j$ is the maximum number of hidden units per hidden group (i.e., associated with one of the variables). This is basically quadratic in the number of variables, like the multi-binomial approximation that uses a polynomial expansion of the joint distribution. However, as $H$ is increased, representation theorems for neural networks suggest that we should be able to approximate with arbitrary precision the true joint distribution. Of course the true limiting factor is the amount of data, and $H$ should be tuned according to the amount of data. In our experiments we have used cross-validation to choose a value of $m_j = H$ for all the hidden groups. In this sense, this neural network representation of $P(Z_1 \ldots Z_n)$ is to the polynomial expansions (such as the multi-binomial) what ordinary multilayer neural networks for function approximation are to polynomial function approximators. It allows to capture high-order dependencies, but not all of them. It is the number of hidden units that controls "how many" such dependencies will be captured, and it is the data that "chooses" which of the actual dependencies are most useful in maximizing the likelihood.

Unlike Bayesian networks with hidden random variables, learning with the proposed architecture is very simple, even when there are no conditional independencies. To optimize the parameters we have simply used gradient-based optimization methods, either using conjugate or stochastic (on-line) gradient, to maximize the total log-likelihood which is the sum of values of $f$ (eq. 2) for the training examples. A prior on the parameters can be incorporated in the cost function and the MAP estimator can be obtained as easily, by maximizing the total log-likelihood plus the log-prior on the parameters. In our experiments we have used a "weight decay" penalty inspired by the analysis of Frey [4], with a penalty proportional to the number of weights incoming into a neuron.

However, it is not so clear how the distribution could be generally marginalized, except by summing over possibly many combinations of the values of variables to be integrated. Another related question is whether one could deal with missing values: if the total number of values that the missing variables can take is reasonably small, then one can sum over these values in order to obtain a marginal probability and maximize this probability. If some variables have more systematically missing values, they can be put at the end of the variable ordering, and in this case it is very easy to compute the marginal distribution (by taking only the product of the output probabilities up to the missing variables). Similarly, one can easily compute the predictive distribution of the last variable given the first $n - 1$ variables.

The framework can be easily extended to hybrid models involving both continuous and discrete variables. In the case of continuous variables, one has to choose a parametric form for the distribution of the continuous variable when all its parents (i.e., the conditioning context) are fixed. For example one could use a normal, log-normal, or mixture of normals. Instead of having softmax outputs, the $i$-th output group would compute the parameters of this continuous distribution (e.g., mean and log-variance). Another type of extension allows to build a conditional distribution, e.g., to model $P(Z_1 \ldots Z_n | X_1 \ldots X_m)$. One just adds extra input units to represent the values of the conditioning variables $X_1 \ldots X_m$. Finally, an architectural extension that we have implemented is to allow direct input-to-output connections (still following the rules of ordering which allow $g_i$ to depend only on $z_1 \ldots z_{i-1}$). Therefore in the case where the number of hidden units is 0 ($H = 0$) we obtain the LARC model proposed by Frey [4].

## 2.2   Choice of topology

Another type of extension of this model which we have found very useful in our experiments is to allow the user to choose a topology that is not fully connected (left-to-right). In our experiments we have used non-parametric tests to heuristically eliminate some of the connections in the network, but one could also use expert or prior knowledge, just as with regular graphical models, in order to cut down on the number of free parameters.

In our experiments we have used for a pairwise test of statistical dependency the Kolmogorov-Smirnov statistic (which works both for continuous and discrete variables). The statistic for variables $X$ and $Y$ is

$$s = \sqrt{l} \sup_i |\hat{P}(X \leq x_i, Y \leq y_i) - \hat{P}(X \leq x_i)\hat{P}(Y \leq y_i)|$$

where $l$ is the number of examples and $\hat{P}$ is the empirical distribution (obtained by counting over the training data). We have ranked the pairs according to their value of the statistic $s$, and we have chosen those pairs for which the value of statistic is above a threshold value $s^*$, which was chosen by cross-validation. When the pairs $\{(Z_i, Z_j)\}$ are chosen to be part of the model, and assuming without loss of generality that $i < j$ for those pairs, then the only connections that are kept in the network (in addition to those from the $k$-th hidden group to the $k$-th output group) are those from hidden group $i$ to output group $j$, and from input group $i$ to hidden group $j$, for every such $(Z_i, Z_j)$ pair.

## 3   Experiments

In the experiments we have compared the following models:

- Naive Bayes: the likelihood is obtained as a product of multinomials (one per variable). Each multinomial is smoothed with a Dirichlet prior.
- Multi-Binomial (using Rademacher-Walsh expansion of order 2) [3]. Since this only handles the case of binary data, it was only applied to the DNA data set.
- A simple graphical model with the same pairs of variables and variable ordering as selected for the neural network, but in which each of the conditional distribution is modeled

by a separate multinomial for each of the conditioning context. This works only if the number of conditioning variables is small so in the Mushroom, Audiology, and Soybean experiments we had to reduce the number of conditioning variables (following the order given by the above tests). The multinomials are also smoothed with a Dirichlet prior.

- Neural network: the architecture described above, with or without hidden units (i.e., LARC), with or without pruning.

5-fold cross-validation was used to select the number of hidden units per hidden group and the weight decay for the neural network and LARC. Cross-validation was also used to choose the amount of pruning in the neural network and LARC, and the amount of smoothing in the Dirichlet priors for the multinomials of the naive Bayes model and the simple graphical model.

## 3.1 Results

All four data sets were obtained on the web from the UCI Machine Learning and STATLOG databases. Most of these are meant to be for classification tasks but we have instead ignored the classification and used the data to learn a probabilistic model of all the input features.

- DNA (from STATLOG): there are 180 binary features. 2000 cases were used for training and cross-validation, and 1186 for testing.
- Mushroom (from UCI): there are 22 discrete features (taking each between 2 and 12 values). 4062 cases were used for training and cross-validation, and 4062 for testing.
- Audiology (from UCI): there are 69 discrete features (taking each between 2 and 7 values). 113 cases are used for training and 113 for testing (the original train-test partition was 200 + 26 and we concatenated and re-split the data to obtain more significant test figures).
- Soybean (from UCI): there are 35 discrete features (taking each between 2 and 8 values). 307 cases are used for training and 376 for testing.

Table 1 clearly shows that the proposed model yields promising results since the pruned neural network was superior to all the other models in all 4 cases, and the pairwise differences with the other models are statistically significant in all 4 cases (except Audiology, where the difference with the network without hidden units, LARC, is not significant).

## 4   Conclusion

In this paper we have proposed a new application of multi-layer neural networks to the modelization of high-dimensional distributions, in particular for discrete data (but the model could also be applied to continuous or mixed discrete / continuous data). Like the polynomial expansions [3] that have been previously proposed for handling such high-dimensional distributions, the model approximates the joint distribution with a reasonable ($O(n^2)$) number of free parameters but unlike these it allows to capture high-order dependencies even when the number of parameters is small. The model can also be seen as an extension of the previously proposed auto-regressive logistic Bayesian network [4], using hidden units to capture some high-order dependencies.

Experimental results on four data sets with many discrete variables are very encouraging. The comparisons were made with a naive Bayes model, with a multi-binomial expansion, with the LARC model and with a simple graphical model, showing that a neural network did significantly better in terms of out-of-sample log-likelihood in all cases.

The approach to pruning the neural network used in the experiments, based on pairwise statistical dependency tests, is highly heuristic and better results might be obtained using approaches that take into account the higher order dependencies when selecting the conditioning variables. Methods based on pruning the fully connected network (e.g., with a "weight elimination" penalty) should also be tried. Also, we have not tried to optimize

| | DNA | | Mushroom | |
|---|---|---|---|---|
| | mean (stdev) | p-value | mean (stdev) | p-value |
| naive Bayes | 100.4 (.18) | <1e-9 | 47.00 (.29) | <1e-9 |
| multi-Binomial order 2 | 117.8 (.01) | <1e-9 | | |
| ordinary graph. model | 108.1 (.06) | <1e-9 | 44.68 (.26) | <1e-9 |
| LARC | 83.2 (.24) | 7e-5 | 42.51 (.16) | <1e-9 |
| pruned LARC | 91.2 (.15) | <1e-9 | 43.87 (.13) | <1e-9 |
| full-conn. neural net. | 120.0 (.02) | <1e-9 | 33.58 (.01) | <1e-9 |
| pruned neural network | 82.9 (.21) | | 31.25 (.04) | |

| | Audiology | | Soybean | |
|---|---|---|---|---|
| | mean (stdev) | p-value | mean (stdev) | p-value |
| naive Bayes | 36.40 (2.9) | <1e-9 | 34.74 (1.0) | <1e-9 |
| multi-Binomial order 2 | | | | |
| ordinary graph. model | 16.56 (.48) | 6.8e-4 | 43.65 (.07) | <1e-9 |
| LARC | 17.69 (.65) | <1e-9 | 16.95 (.35) | 5.5e-4 |
| pruned LARC | 16.69 (.41) | 0.20 | 19.06 (.43) | <1e-9 |
| full-conn. neural net. | 17.39 (.58) | <1e-9 | 21.65 (.43) | <1e-9 |
| pruned neural network | 16.37 (.45) | | 16.55 (.27) | |

Table 1: *Average out-of-sample negative log-likelihood obtained with the various models on four data sets (standard deviations of the average in parenthesis and p-value to test the null hypotheses that a model has same true generalization error as the pruned neural network). The pruned neural network was better than all the other models in in all cases, and the pair-wise difference is always statistically significant (except with respect to the pruned LARC on Audiology).*

the order of the variables, or combine different networks obtained with different orders, like [4].

## Footnotes

[0]Part of this work was done while S.B. was at CIRANO, Montreal, Qc. Canada.

## References

[1] R.R. Bahadur. A representation of the joint distribution of responses to n dichotomous items. In ed. H. Solomon, editor, *Studies in Item Analysis and Predictdion*, pages 158–168. Stanford University Press, California, 1961.

[2] C.K. Chow. A recognition method using neighbor dependence. *IRE Trans. Elec. Comp.*, EC-11:683–690, October 1962.

[3] R.O. Duda and P.E. Hart. *Pattern Classification and Scene Analysis*. Wiley, New York, 1973.

[4] B. Frey. *Graphical models for machine learning and digital communication*. MIT Press, 1998.

[5] Steffen L. Lauritzen. The EM algorithm for graphical association models with missing data. *Computational Statistics and Data Analysis*, 19:191–201, 1995.

[6] Judea Pearl. *Probabilistic Reasoning in Intelligent Systems : Networks of Plausible Inference*. Morgan Kaufmann, 1988.